# Two-Dimensional Object Localization by Coarse-to-Fine Correlation Matching

**Chien-Ping Lu and Eric Mjolsness**
Department of Computer Science
Yale University
New Haven, CT 06520-8285

## Abstract

We present a Mean Field Theory method for locating two-dimensional objects that have undergone rigid transformations. The resulting algorithm is a form of coarse-to-fine correlation matching. We first consider problems of matching synthetic point data, and derive a point matching objective function. A tractable line segment matching objective function is derived by considering each line segment as a dense collection of points, and approximating it by a sum of Gaussians. The algorithm is tested on real images from which line segments are extracted and matched.

## 1 Introduction

Assume that an object in a scene can be viewed as an instance of the model placed in space by some spatial transformation, and object recognition is achieved by discovering an instance of the model in the scene. Two tightly coupled subproblems need to be solved for locating and recognizing the model: *the correspondence problem* (how are scene features put into correspondence with model features?), and *the localization problem* (what is the transformation that acceptably relates the model features to the scene features?). If the correspondence is known, the transformation can be determined easily by least squares procedures. Similarly, for known transformation, the correspondence can be found by aligning the model with the scene, or the problem becomes an assignment problem if the scene feature locations are jittered by noise.

Several approaches have been proposed to solve this problem. Some *tree-pruning methods* [1, 3] make hypotheses concerning the correspondence by searching over a tree in which each node represents a partial match. Each partial match is then evaluated through the pose that best fits it. In the *generalized Hough transform* or equivalently *template matching* approach [7, 3], optimal transformation parameters are computed for each possible pairing of a model feature and a scene feature, and these "optimal" parameters then "vote" for the closest candidate in the discretized transformation space.

By contrast with the tree-pruning methods and the generalized Hough transform, we propose to formulate the problem as an objective function and optimize it directly by using Mean Field Theory (MFT) techniques from statistical physics, adapted as necessary to produce effective algorithms in the form of analog neural networks.

## 2   Point Matching

Consider the problem of locating a two-dimensional "model" object that is believed to appear in the "scene". Assume first that both the model and the scene are represented by a set of "points" respectively, $\{\mathbf{x}_i\}$ and $\{\mathbf{y}_a\}$. The problem is to recover the actual transformation (translation and rotation) that relates the two sets of points. It can be solved by minimizing the following objective function

$$E_{\text{match}}(\mathbf{M}_{ia}, \theta, \mathbf{t}) = \sum_{ia} M_{ia} \|\mathbf{x}_i - \mathbf{R}_\theta \mathbf{y}_a - \mathbf{t}\|^2 \tag{1}$$

where $\{M_{ia}\} = \mathbf{M}$ is a 0/1-valued "match matrix" representing the unknown correspondence, $\mathbf{R}_\theta$ is a rotation matrix with rotation angle $\theta$, and $\mathbf{t}$ is a translation vector.

### 2.1   Constraints on match variables

We need to enforce some constraints on correspondence (match) variables $M_{ia}$; otherwise all $M_{ia} = 0$ in (1). Here, we use the following constraint

$$\sum_{ia} M_{ia} = N, \ \forall M_{ia} \geq 0; \tag{2}$$

implying that there are exactly $N$ matches among all possible matches, where $N$ is the number of the model features. Summing over permutation matrices obeying this constraint, the effective objective function is approximately [5]:

$$F(\theta, \mathbf{t}, \beta) = -\frac{1}{\beta} \sum_{ia} e^{-\beta \|\mathbf{x}_i - \mathbf{R}_\theta \mathbf{y}_a - \mathbf{t}\|^2}, \tag{3}$$

which has the same fixed points as

$$E_{\text{penalty}}(\mathbf{M}, \theta, \mathbf{t}) = E_{\text{match}}(\mathbf{M}, \theta, \mathbf{t}) + \frac{1}{\beta} \sum_{ia} M_{ia}(\log M_{ia} - 1), \tag{4}$$

where $M_{ia}$ is treated as a continuous variable and is subject to the penalty function $x(\log x - 1)$.

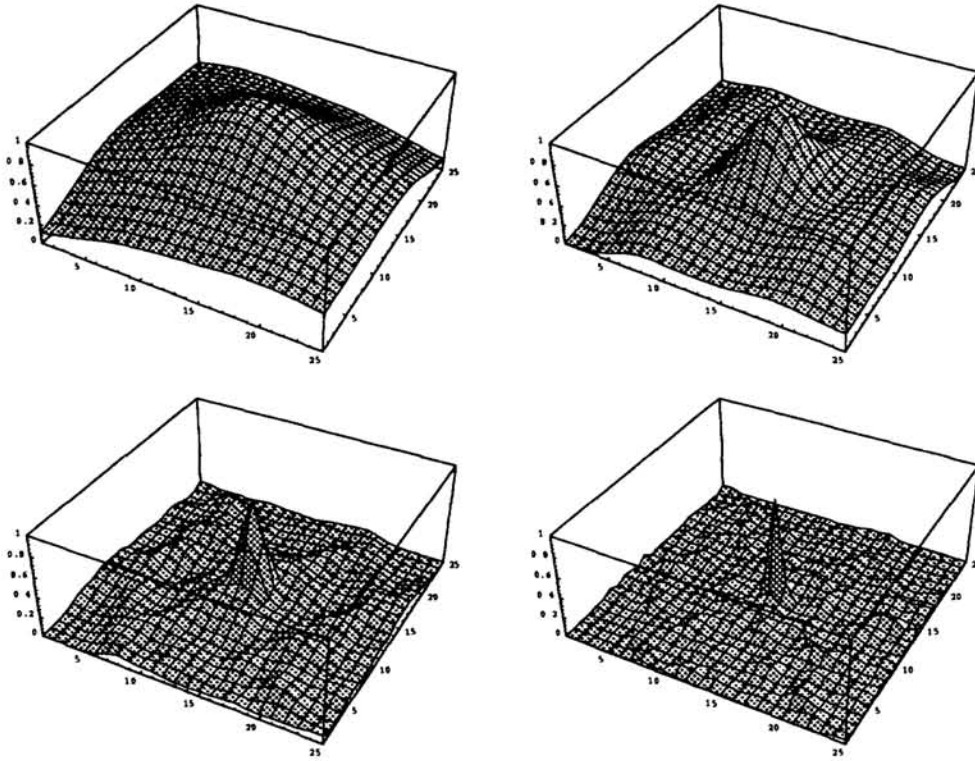

Figure 1: Assume that there is only translation between the model and the scene, each containing 20 points. The objective functions at at different temperatures ($\beta^{-1}$): 0.0512 (*top left*), 0.0128 (*top right*), 0.0032 (*bottom left*) and 0.0008 (*bottom right*), are plotted as energy surfaces of $x$ and $y$ components of translation.

Now, let $\beta = 1/2\sigma^2$ and write

$$E_{\text{point}}(\theta, \mathbf{t}) = \sum_{ia} e^{-\frac{1}{2\sigma^2}\|\mathbf{x}_i - \mathbf{R}_\theta \mathbf{y}_a - \mathbf{t}\|^2}. \tag{5}$$

The problem then becomes that of maximizing $E_{\text{point}}$, which in turn can be interpretated as minimizing the Euclidean distance between two Gaussian-blurred images containing the scene points $\mathbf{x}_i$ and a transformed version of the model points $\mathbf{y}_a$. Tracking the local maximum of the objective function from large $\sigma$ to small $\sigma$, as in deterministic annealing and other continuation methods, corresponds to a coarse-to-fine correlation matching. See Figure 1 for a demonstration of a simpler case in which only translation is applied to the model.

## 2.2   The descent dynamics

A gradient descent dynamics for finding the saddle point of the effective objective function $F$ is

$$\dot{\mathbf{t}} = -\kappa \sum_{ia} m_{ia}(\mathbf{x}_i - \mathbf{R}_\theta \mathbf{y}_a - \mathbf{t})$$

$$\dot{\theta} = -\kappa \sum_{ia} m_{ia}(\mathbf{x}_i - \mathbf{R}_\theta \mathbf{y}_a - \mathbf{t})^t (\mathbf{R}_{\theta + \frac{\pi}{2}} \mathbf{y}_a), \tag{6}$$

where $m_{ia} = \langle M_{ia} \rangle_\beta = e^{-\beta \|\mathbf{x}_i - \mathbf{R}_\theta \mathbf{y}_a - \mathbf{t}\|^2}$ is the "soft correspondence" associated with $M_{ia}$. Instead of updating $\mathbf{t}$ by descent dynamics, we can also solve for $\mathbf{t}$ directly.

## 3   The Vernier Network

Though the effective objective is non-convex over translation at low temperatures, its dependence on rotation is non-convex even at relatively high temperatures.

### 3.1   Hierachical representation of variables

We propose overcoming this problem by applying Mean Field Theory (MFT) to a hierachical representation of rotation resulting from the change of variables [4]

$$\theta = \sum_{b=0}^{B-1} \chi_b (\hat{\theta}_b + \theta_b), \ \theta_b \in [-\epsilon_\theta, \epsilon_\theta], \tag{7}$$

where $\epsilon_\theta = \pi/2B$, $\hat{\theta}_b = (b + \frac{1}{2})\frac{\pi}{B}$ are the constant centers of the intervals, and $\theta_b$ are fine-scale "vernier" variables. The $\chi_b$'s are binary variables (so $\chi_b \in \{0, 1\}$) that satisfy the winner-take-all (WTA) constraint $\sum_b \chi_b = 1$.

The essential reason that this hierarchical representation of $\theta$ has fewer spurious local minima than the conventional analog representation is that the change of variables also changes the *connectivity of the network's state space*: big jumps in $\theta$ can be achieved by local variations of $\chi$.

### 3.2   Vernier optimization dynamics

$E_{\text{point}}$ can be transformed as (see [6, 4]) [1]

$$E_{\text{point}}(\theta, \mathbf{t}) \xrightarrow{\Delta \text{vbl}} E\left(\sum_b \chi_b(\hat{\theta}_b + \theta_b), \sum_b \chi_b \mathbf{t}_b\right)$$

$$= \sum_b \chi_b E(\hat{\theta}_b + \theta_b, \mathbf{t}_b)$$

$$\sum_\alpha \Psi_\alpha(t) E(\{z\}_\alpha, \{\bar{z}\}_\alpha) \equiv E(x, \bar{y}) \oplus E(\bar{x}, y) \equiv E\langle x, y \rangle_\oplus \tag{8}$$

- $\oplus$ for clocked sum
- $\bar{x}$ for a clamped variable
- $x^A$ for a set of variables to be optimized analytically
- $(v, u)^H$ for Hopfield/Grossberg dynamics
- $E\langle x, y \rangle_\oplus$ for coordinate descent/ascent on $x$, then $y$, iterated if necessary. Nested angle brackets correspond to nested loops.

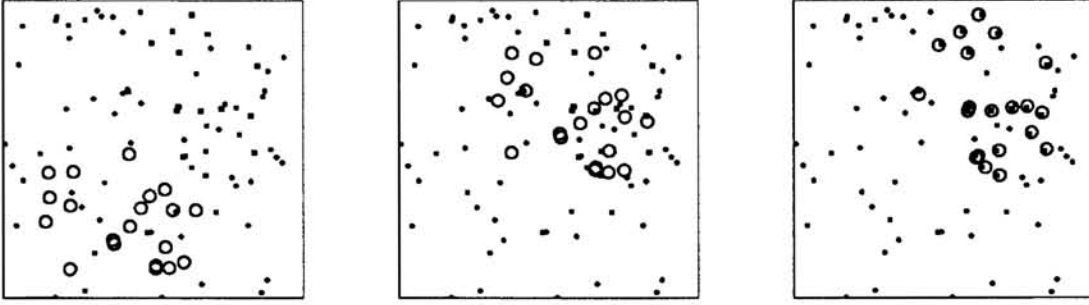

Figure 2: Shown here is an example of matching a 20-point model to a scene with 66.7% spurious outliers. The model is represented by circles. The set of square dots is an instance of the model in the scene. All other dots are outliers. From left to right are configurations at the annealing steps 1, 10, and 51, respectively.

$$\stackrel{\text{MFT}}{\rightarrow} \left[ \sum_b \chi_b E(\hat{\theta}_b + v_b, \mathbf{t}_b) + \frac{1}{\beta} \sum_b (u_b v_b - \log \frac{\sinh(\epsilon u_b)}{\epsilon}) \right.$$
$$\left. + \text{WTA}(\chi, \beta) \right] \langle\langle (\mathbf{v}, \mathbf{u})^H, \mathbf{t}^A \rangle, \chi^A \rangle_\oplus \qquad (9)$$

Each bin-specific rotation angle $v_b$ can be found by the following fixed point equations

$$\mathbf{t}_b = \sum_a m_{ia}(\mathbf{x}_i - \mathbf{R}_{v_b}\mathbf{y}_a)$$

$$u_b = -\beta \sum_{ia} m_{ia}(\mathbf{x}_i - \mathbf{R}_{v_b}\mathbf{y}_a - \mathbf{t}_b)^t(\mathbf{R}_{v_b + \frac{\pi}{2}}\mathbf{y}_a)$$

$$v_b = \langle \theta_b + \hat{\theta}_b \rangle_\beta = \frac{1}{u_b} - \frac{\epsilon_\theta}{\tanh(\epsilon_\theta u_b)} = g(u_b). \qquad (10)$$

The algorithm is illustrated in Figure 2.

## 4  Line Segment Matching

In many vision problems, representation of images by line segments has the advantage of compactness and subpixel accuracy along the direction transverse to the line. However, such a representation of an object may vary substantially from image to image due to occlusions and different illumination conditions.

### 4.1  Indexing points on line segements

The problem of matching line segments can be thought of as a point matching problem in which each line segment is treated as a dense collection of points. Assume now that both the scene and the model are represented by a set of line segments respectively, $\{\mathbf{s}_i\}$ and $\{\mathbf{m}_a\}$. Both the model and the scene line segments are

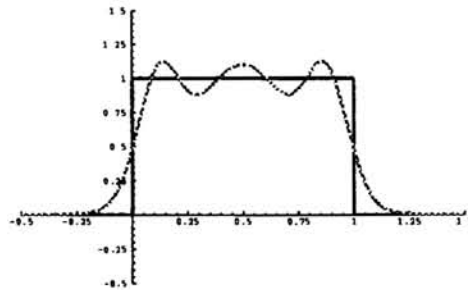

Figure 3: Approximating $\Theta(t)$ by a sum of 3 Gaussians.

represented by their endpoints as $\mathbf{s}_i = (\mathbf{p}_i, \mathbf{p}_i')$ and $\mathbf{m}_a = (\mathbf{q}_a, \mathbf{q}_a')$, where $\mathbf{p}_i, \mathbf{p}_i'$, and $\mathbf{q}_a, \mathbf{q}_a'$ are the endpoints of the $i$th scene segment and the $a$th model segment, respectively. The locations of the points on each scene segment and model segments can be parameterized as

$$\mathbf{x_i} = \mathbf{s}_i(u) = \quad \mathbf{p}_i + u(\mathbf{p}_i' - \mathbf{p}_i), \quad u \in [0,1] \text{ and} \tag{11}$$

$$\mathbf{y_a} = \mathbf{m}_a(v) = \quad \mathbf{q}_a + v(\mathbf{q}_a' - \mathbf{q}_a), \quad v \in [0,1]. \tag{12}$$

Now the model points and the scene points can be though of as indexed by $\mathbf{i} = (i, u)$ and $\mathbf{a} = (a, v)$. Using this indexing, we have $\sum_{\mathbf{i}} \propto \sum_i l_i \int_0^1 du$ and $\sum_{\mathbf{a}} \propto \sum_a l_a \int_0^1 dv$, where $l_i = \|\mathbf{p}_i - \mathbf{p}_i'\|$ and $l_a = \|\mathbf{q}_a - \mathbf{q}_a'\|$. The point matching objective function (5) can be specialized to line segment matching as [5]

$$E_{\text{seg}}(\theta, \mathbf{t}) = \sum_{ia} l_i l_a \int_0^1 \int_0^1 e^{-\frac{1}{2\sigma^2}\|\mathbf{s}_i(u) - \mathbf{R}_\theta \mathbf{m}_a(v) - \mathbf{t}\|^2} du \, dv. \tag{13}$$

As a special case of point matching objective function, (13) can readily be transformed to the vernier network previously developed for point matching problem.

### 4.2   Gaussian sum approximation

Note that, as in Figure 3 and [5],

$$\Theta(t) \equiv \left\{ \begin{array}{ll} 1 & \text{if } t \in [0,1] \\ 0 & \text{otherwise} \end{array} \right. \approx \sum_{k=1}^{3} A_k \exp -\frac{1}{2} \frac{(c_k - t)^2}{\sigma_k^2} \tag{14}$$

where by numerical minimization of the Euclidean distance between these two functions of $t$, the parameters may be chosen as $A_1 = A_3 = 0.800673$, $A_2 = 1.09862$, $\sigma_1 = \sigma_3 = 0.0929032$, $\sigma_2 = 0.237033$, $c_1 = 1 - c_3 = 0.116807$, and $c_2 = 0.5$.

Using this approximation, each finite double integral in (13) can be replaced by

$$\sum_{k,l=1}^{3} A_k A_l \int_{-\infty}^{+\infty} \int_{-\infty}^{+\infty} e^{-\frac{1}{2\sigma_k^2}(c_k-u)^2} e^{-\frac{1}{2\sigma_l^2}(c_l-v)^2} e^{-\frac{1}{2\sigma^2}\|\mathbf{s}_i(u)+\mathbf{R}_\theta \mathbf{m}_a(v)-\mathbf{t}\|^2} du \, dv. \tag{15}$$

Each of these nine Gaussian integrals can be done exactly. Defining

$$\mathbf{v}_{iakl} = \mathbf{s}_i(c_k) - \mathbf{R}_\theta \mathbf{m}_a(c_l) - \mathbf{t} \tag{16}$$

$$\hat{\mathbf{p}}_i = \mathbf{p}_i' - \mathbf{p}_i, \quad \hat{\mathbf{q}}_a = \mathbf{R}_\theta(\mathbf{q}_a' - \mathbf{q}_a), \tag{17}$$

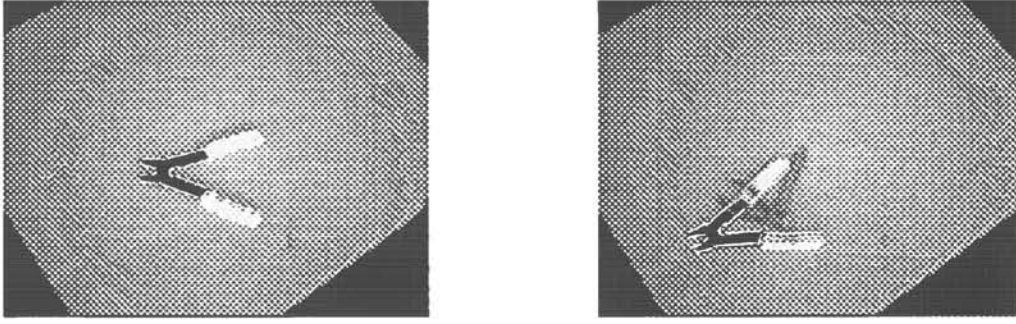

Figure 4: The model line segments, which are transformed with the optimal parameter found by the matching algorithm, are overlayed on the scene image. The algorithm has successfully located the model object in the scene.

(15) becomes

$$
2\pi\sigma^2 l_i l_a \sum_{k,l=1}^{3} \frac{A_k A_l \sigma_k \sigma_l}{\sqrt{(\sigma^2 + \hat{\mathbf{p}}_i^2 \sigma_k^2)(\sigma^2 + \hat{\mathbf{q}}_a^2 \sigma_l^2) - \sigma_i^2 \sigma_j^2 (\hat{\mathbf{p}}_i \cdot \hat{\mathbf{q}}_a)^2}}
$$
$$
\times \exp -\frac{1}{2} \frac{\mathbf{v}_{iakl}^2 \sigma^2 + (\mathbf{v}_{iakl} \times \hat{\mathbf{p}}_i)^2 \sigma_k^2 + (\mathbf{v}_{iakl} \times \hat{\mathbf{q}}_a)^2 \sigma_l^2}{(\sigma^2 + \hat{\mathbf{p}}_i^2 \sigma_k^2)(\sigma^2 + \hat{\mathbf{q}}_a^2 \sigma_l^2) - \sigma_k^2 \sigma_l^2 (\hat{\mathbf{p}}_i \cdot \hat{\mathbf{q}}_a)^2} \tag{18}
$$

as was calculated by Garrett [2, 5]. From the Gaussian sum approximation, we get a closed form objective function which can be readily optimized to give a solution to the line segment matching problem.

## 5   Results and Discussion

The line segment matching algorithm described in this paper was tested on scenes captured by a CCD camera producing $640 \times 480$ images, which were then processed by an edge detector. Line segments were extracted using a polygonal approximation to the edge images. The model line segments were extracted from a scene containing a canonically positioned model object (Figure 4 *left*). They were then matched to that extracted from a scene containing differently positioned and partially occluded model object (Figure 4 *right*). The result of matching is shown in Figure 5.

Our approach is based on a scale-space continuation scheme derived from an application of Mean Field Theory to the match variables. It provides a means to avoid trapping by local extrema and is more efficient than stochastic searches such as simulated annealing. The estimation of location parameters based on continuously improved "soft correspondences" and scale-space is often more robust than that based on crisp (but usually inaccurate) correspondences.

The vernier optimization dynamics arises from an application of Mean Field Theory to a hierarchical representation of the rotation, which turns the original unconstrained optimization problem over rotation $\theta$ into several constrained optimization problems over smaller $\theta$ intervals. Such a transformation results in a Hopfield-style

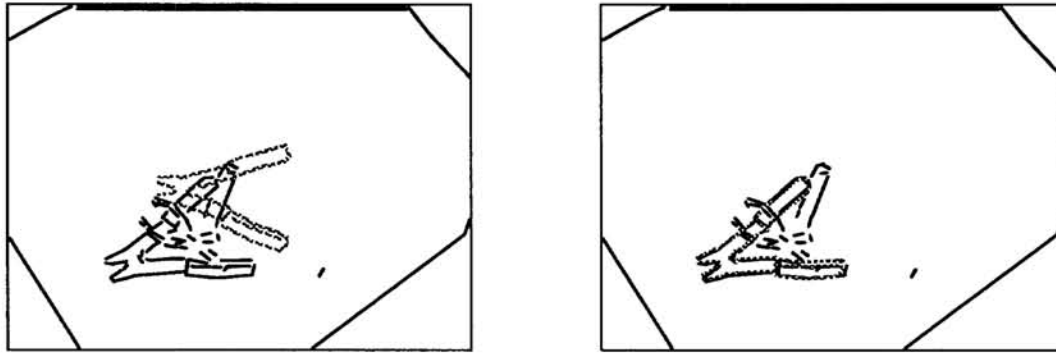

Figure 5: Shows how the model line segments (gray) and the scene segments (black) are matched. The model line segments, which are transformed with the optimal parameter found by the matching algorithm, are overlayed on the scene line segments with which they are matched. Most of the the endpoints and the lengths of the line segments are different. Furthermore, one long segment frequently corresponds to several short ones. However, the matching algorithm is robust enough to uncover the underlying rigid transformation from the incomplete and ambiguous data.

dynamics on rotation $\theta$, which effectively coordinates the dynamics of rotation and translation during the optimization. The algorithm tends to find a roughly correct translation first, and then tunes up the rotation.

## 6   Acknowledgements

This work was supported under grant N00014-92-J-4048 from ONR/DARPA.

## Footnotes

[1] Notation: Coordinate descent with 2-phase clock $\Psi_\alpha(t)$:

## References

[1] H. S. Baird. *Model-Based Image Matching Using Location.* The MIT Press, Cambridge, Massachusetts, first edition, 84.

[2] C. Garrett, 1990. Private communication to Eric Mjolsness.

[3] W. E. L. Grimson and T. Lozano-Perez. Localizing overlapping parts by searching the interpretation tree. *IEEE Transaction on Pattern Analysis and Machine Intelligence*, 9:469–482, 1987.

[4] C.-P. Lu and E. Mjolsness. Mean field point matching by vernier network and by generalized Hough transform. In *World Congress on Neural Networks*, pages 674–684, 1993.

[5] E. Mjolsness. Bayesian inference on visual grammars by neural nets that optimize. In *SPIE Science of Artificial Neural Networks*, pages 63–85, April 1992.

[6] E. Mjolsness and W. L. Miranker. Greedy Lagrangians for neural networks: Three levels of optimization in relaxation dynamics. Technical Report YALEU/DCS/TR-945, Yale Computer Science Department, January 1993.

[7] G. Stockman. Object recognition and localization via pose clustering. *Computer Vision, Graphics, and Image Processing*, (40), 1987.